# Utilizing Time: Asynchronous Binding

**Bradley C. Love**
Department of Psychology
Northwestern University
Evanston, IL 60208

## Abstract

Historically, connectionist systems have not excelled at representing and manipulating complex structures. How can a system composed of simple neuron-like computing elements encode complex relations? Recently, researchers have begun to appreciate that representations can extend in both time and space. Many researchers have proposed that the synchronous firing of units can encode complex representations. I identify the limitations of this approach and present an *asynchronous* model of binding that effectively represents complex structures. The asynchronous model extends the synchronous approach. I argue that our cognitive architecture utilizes a similar mechanism.

## 1    Introduction

Simple connectionist models can fall prey to the "binding problem". A binding problem occurs when two different events (or objects) are represented identically. For example, representing "John hit Ted" by activating the units **JOHN**, **HIT**, and **TED** would lead to a binding problem because the same pattern of activation would also be used to represent "Ted hit John". The binding problem is ubiquitous and is a concern whenever internal representations are postulated. In addition to guarding against the binding problem, an effective binding mechanism must construct representations that assist processing. For instance, different states of the world must be represented in a manner that assists in discovering commonalities between disparate states, allowing for category formation and analogical processing.

Interestingly, new connectionist binding mechanisms [5, 9, 12] utilize time in their operation. Pollack's Recursive Auto-Associative Memory (RAAM) model combines a standard fixed-width multi-layer network architecture with a stack and a simple controller, enabling RAAM to encode hierarchical representations over multiple processing steps. RAAM requires more time to encode representations as they become more complex, but its space requirements remain constant. The clearest example

of utilizing time are models that perform dynamic binding through synchronous firings of units [17, 5, 12]. Synchrony models explicitly use time to mark relations between units, distributing complex representations across multiple time steps.

Most other models neglect the time aspect of representation. Even synchrony models fail to fully utilize time (I will clarify this point in a later section). In this paper, a model is introduced (the asynchronous binding mechanism) that attempts to rectify this situation. The asynchronous approach is similar to the synchronous approach but is more effective in binding complex representations and exploiting time.

## 2   Utilizing time and the brain

Representational power can be greatly increased by taking advantage of the time dimension of representation. For instance, a telephone would need thousands of buttons to make a call if sequences of digits were not used. From the standpoint of a neuron, taking advantage of timing information increases processing capacity by more than a 100 fold [13]. While this suggests that the neural code might utilize both time and space resources, the neuroscience community has not yet arrived at a consensus. While it is known that the behavior of a postsynaptic neuron is affected by the location and arrival times of dendritic input [10], it is generally believed that only the rate of firing (a neuron's firing rate is akin to the activation level of a unit in a connectionist network) can code information, as opposed to the timing of spikes, since neurons are noisy devices [14]. However, findings that are taken as evidence for rate coding, like elevated firing rates in memory retention tasks [8], can often be reinterpreted as part of complex cortical events that extend through time [1]. In accord with this view, recent empirical findings suggests that the timing of spikes (e.g., firing patterns, intervals) are also part of the neural code [4, 16]. Contrary to the rate based view (which holds only that only the firing rate of a neuron encodes information), these studies suggest that the timing of spikes encodes information (e.g., when two neurons repeatedly spike together it signifies something different than when they fire out of phase, even if their firing rates are identical in both cases).

Behavioral findings also appear consistent with the idea that time is used to construct complex representations. Behavioral research in illusory conjunction phenomena [15], and sentence processing performance [11] all suggest that bindings or relations are established through time, with bindings becoming more certain as processing proceeds. In summary, early in processing humans can gauge which representational elements are relevant while remaining uncertain about how these elements are interrelated.

## 3   Dynamic binding through synchrony

Given the demands placed on a representational system, a system that utilizes dynamic binding through synchrony would seem to be a good candidate mental architecture (though, as we will see, limitations arise when representing complex structures). A synchronous binding account of our mental architecture is consistent (at a general level) with behavioral findings, the intuition that complex representations are distributed across time, and that neural temporal dynamics code information. Synchrony seems to offer the power to recombine a finite set of elements in a virtually unlimited number of ways (the defining characteristic of a discrete combinatorial system).

While synchrony models seem appropriate for modeling certain behaviors, dynamic binding through synchrony does not seem to be an appropriate mechanism for establishing complex recursive bindings [2]. In a synchronous dynamic binding system, the distinction between a slot and a filler is lost, since bindings are not directional (i.e., which unit is a predicate and which unit is an argument is not clear). The slot and the filler simply share the same phase. In this sense, the mechanism is more akin to a grouping mechanism than to a binding mechanism. Grouping units together indicates that the units are a part of the same representation, but does not sort out the relations among the units as binding does.

Synchrony runs into trouble when a unit has to act simultaneously as a slot and a filler. For instance, to represent embedded propositions with synchronous binding, a controller needs to be added. For instance, a structure with embedding, like $A \rightarrow B \rightarrow C$, could be represented with synchronous firings if $A$ and $B$ fired synchronously and then $B$ and $C$ fired synchronously. Still, synchronous binding blurs the distinction between a slot and a filler, necessitating that $A$, $B$, and $C$ be marked as slots or fillers to unambiguously represent the simple $A \rightarrow B \rightarrow C$ structure. Notice that $B$ must be marked as a slot when it fires synchronously with $A$, but must be marked as filler when it synchronously fires with $C$. When representing embedded structures, the synchronous approach becomes complicated (i.e., simple connections are not sufficient to modulate firing patterns) and rigid (i.e., parallelism and flexibility are lost when a unit has to be either a slot or a filler). Ideally, units would be able to act simultaneously as slots and fillers, instead of alternating between these two structural roles.

## 4    The asynchronous approach

While synchrony models utilize some timing information, other valuable timing information is discarded as noise, making it difficult to represent multiple levels of structure. If $A$ fired slightly before $B$, which fired slightly before $C$, asynchronous timing information (ordering information) would be available. This ordering information allows for directional binding relations and alleviates the need to label units as slots or fillers. Notice that $B$ can act simultaneously as a slot and a filler. Directional bindings can unambiguously represent complex structures.

Phase locking and wave like patterns of firing need not occur during asynchronous binding. For instance, the firing pattern that encodes a structure like $A \rightarrow B \rightarrow C$ does not need to be orderly (i.e., starting with $A$ and ending with $C$). To encode $A \rightarrow B \rightarrow C$, unit $B$'s firing schedule must observably speed up (on average) after unit $A$ fires, while $C$'s must speed up after $B$ fires. For example, if we only considered the time window immediately after a unit fires, a firing sequence of $B$, $C$, no unit fires, $A$, and then $B$ would provide evidence for the structure $A \rightarrow B \rightarrow C$. Of course, if $A$, $B$, and $C$ fire periodically with stochastic schedules that are influenced by other units' firings, spurious binding evidence will accrue (e.g., occasionally, $C$ will fire and $A$ will fire in the next time step). Luckily, these accidents will be less frequent than events that support the intended bindings. As binding evidence is accumulated over time, binding errors will become less likely.

Interestingly, the asynchronous mechanism can also represent structures through an inhibitory process that mirrors the excitatory process described above. $A \rightarrow B \rightarrow C$ could be represented asynchronously if $A$ was less likely to fire after $B$ fired and $B$ was less likely to fire after $C$ fired. An inhibitory (negative) connection from $B$ to $A$ is in some ways equivalent to an excitatory (positive) connection form $A$ to $B$.

## 4.1 The mathematical expression of the model

The previous discussion of the asynchronous approach can be formalized. Below is a description of an asynchronous model that I have implemented.

### 4.1.1 The anatomy of a unit

Individual units, when unaffected by other units, will fire periodically when active:

$$\text{if } R_{t_i} \geq 1, \text{ then } O_{t_{i+1}} = 1, \text{ otherwise } O_{t_{i+1}} = 0. \tag{1}$$

where $O_{t_{i+1}}$ is the unit's output (at time $i + 1$), $R_{t_i}$ is the unit's output refractory period which is randomly set (after the unit fires) to a value drawn from the uniform distribution between 0 and 1 and is incremented at each time step by some constant (which was set to .1 in all simulations). Notice that a unit produces an output one time step after its output refractory period reaches threshold.

### 4.1.2 A unit's behavior in the presence of other units

A unit alters its output refractory if it receives a signal (via a connection) from a unit that has just fired (i.e., a unit with a positive output). For example, if unit **A** fires (its output is 1) and there is a connection to unit **B** of strength +.3, then **B**'s output refractory will be incremented by +.3, enabling unit **B** to fire during the next time step or at least decreasing the time until **B** fires. Alternatively, negative (inhibitory) connections lower refractory.

Two unconnected units will tend to fire independently of each other, providing little evidence for a binding relation. Again, over a small time window, two units may fire contiguously by chance, but over many firings the evidence for a binding will approach zero.

### 4.1.3 Interpreting firing patterns

Every time a unit fires, it creates evidence for binding hypotheses. The critical issue is how to collect and evaluate evidence for bindings. There are many possible evidence functions that interpret firing patterns in a sensible fashion. One simple function is to have evidence for two units binding decrease linearly as the time between their firings increases. Evidence is updated every time step according to the following equation:

$$\text{if } \rho \geq \left(t_{U_j} - t_{U_i}\right) \geq 1, \text{ then } \Delta E_{ij} = -\left(1/\rho\right)\left(t_{U_j} - t_{U_i}\right) + (1/\rho) + 1. \tag{2}$$

where $\rho$ is the size of the window for considering binding evidence (i.e., if $\rho$ is 5, then units firing 5 time steps apart still generate binding evidence), $t_{U_i}$ is the most recent time step unit $U_i$ fired, and $\Delta E_{ij}$ is the change in the amount of evidence for $U_i$ binding to $U_j$. Of course, some evidence will be spurious. The following decision rule can be used to determine if two units share a binding relation:

$$\text{if } (E_{ij} - E_{ji}) > k, \text{ then } U_i \text{ binds to } U_j. \tag{3}$$

where $k$ is some threshold greater than 0. This decision rule is formally equivalent to the diffusion model which is a type of random walk model [6]. Equations 2 and 3 are very simple. Other more sophisticated methods can be used for collecting and evaluating binding evidence.

## 4.2 Performance of the Asynchronous Mechanism

In this section, the asynchronous binding mechanism's performance characteristics are examined. In particular, the model's ability to represent tree structures of

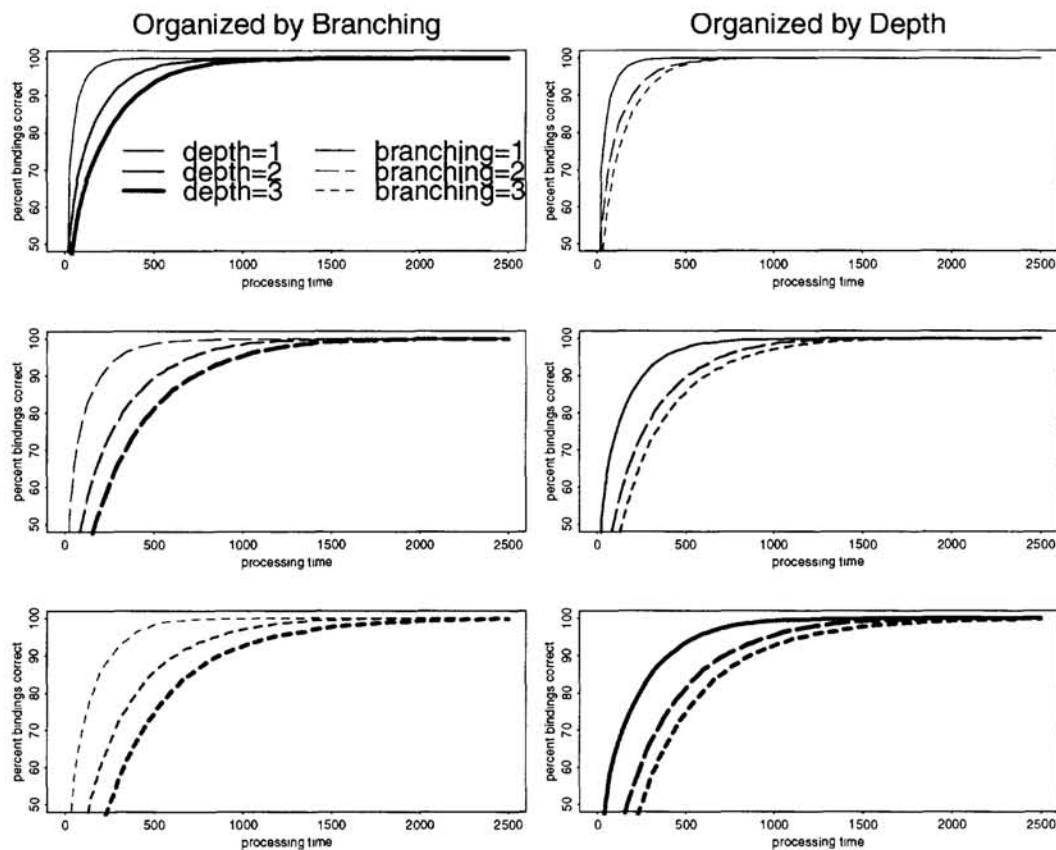

Figure 1: Performance curves for the 9 different structures are shown.

varying complexity was explored. Tree structures can be used to represent complex relational information, like the parse of a sentence. An advantage of using tree structures to measure performance is that the complexity of a tree can be easily described by two factors. Trees can vary in their depth and branching. In the simulations reported here, trees had a branching factor and depth of either 1, 2, or 3. These two factors were crossed, yielding 9 different tree structures. This design makes it possible to assess how the model processes structures of varying complexity. One sensible prediction (given our intuitions about how we process structured representations) is that trees with greater depth and branching will take longer to represent.

In the simulations reported here, both positive and negative connections were used simultaneously. For instance, in a tree structure, if **A** was intended to bind to **B**, **A**'s connection to **B** was set to +.1 and **B**'s connection to **A** was set to −.1. The combination of both connection types yields the best performance.

In these simulations both excitatory and inhibitory binding connection values were set relatively low (all binding connections were set to size .1), providing a strict test of the model's sensitivity. The low connection values prevented bound units from establishing tight couplings (characteristic of bound units in synchrony models). For example, with an excitatory connection from **A** to **B** of .1, **A**'s firing does not ensure that **B** will fire in the next time step (or the next few time steps for that matter). The lack of a tight coupling requires the model to be more sensitive to how one unit affects another unit's firing schedule. With all connections of size .1, firing patterns representing complex structures will appear chaotic and unorderly.

In all simulations, the time window for considering binding evidence was 5 time steps (i.e., Equation 2 was used with $\rho$ set to 5).

Performance was measured by calculating the percent bindings correct. The bindings the model settled upon were determined by calculating the number of bindings in the intended structure. The model then created a structure with this number of bindings (this is equivalent to treating $k$ like a free parameter), choosing the bindings it believed to be most likely (based on accrued evidence). The model was correct when the bindings it believed to be present corresponded to the intended bindings.

For each of the 9 structures (3 levels of depth by 3 levels of branching), hundreds of trials were run (the mechanism is stochastic) until performance curves became smooth. The model's performance was measured every 25th time step up to the 2500th time step. Performance (averaged across trials) for all structures is shown in Figure 1. Any viewable difference between performance curves is statistically significant. As predicted, there was a main effect for both branching and depth. The left panels of Figure 1 organize the data by branching factor, revealing a systematic effect of depth. The right panel is organized by depth and reveals a systematic effect of branching. As structures become more complex, they appear to take longer to represent.

## 5 Conclusions

The ability to effectively represent and manipulate complex knowledge structures is central to human cognition [3]. Connectionists models generally lack this ability, making it difficult to give a connectionist account of our mental architecture. The asynchronous mechanism provides a connectionist framework for representing structures in a way that is biologically, computationally, and behaviorally feasible. The mechanism establishes bindings over time using simple neuron-like computing elements. The asynchronous approach treats bindings as directional and does not blur the distinction between a slot and a filler as the synchronous approach does.

The asynchronous mechanism builds representations that can be differentiated from each other, capturing important differences between representational states. The representations that the asynchronous mechanism builds also can be easily compared and commonalities between disparate states can be extracted by analogical processes, allowing for generalization and feature discovery. In fact, an analogical (i.e., graph) matcher has been built using the asynchronous mechanism [7]. Variants of the model need to be explored. This paper only outlines the essentials of the architecture. Synchronous dynamic binding models were partly inspired from work in neuroscience. Hopefully the asynchronous dynamic binding model will now inspire neuroscience researchers. Some evidence for rate-based firing (spatially based) neural codes has been revisited and viewed as consistent with more complex temporal codes [1]; perhaps evidence for synchrony can be subjected to more sophisticated analyses and be better construed as evidence for the asynchronous mechanism.

**Acknowledgments**

This work was supported by the Office of Naval Research under the National Defense Science and Engineering Graduate Fellowship Program. I would like to thank John Hummel for his helpful comments.

# References

[1] M. Abeles, H. Bergman, E. Margalit, and E. Vaadia. Spatiotemporal firing patterns in the frontal cortex of behaving monkeys. *Journal of Neurophysiology*, 70:1629–1638, 1993.

[2] E. Bienenstock. Composition. In A. Aertsen and V. Braitenberg, editors, *Brain Theory: Biological Basis and Computational Principles*. Elsevier, New York, 1996.

[3] D. Gentner and A. B. Markman. Analogy-watershed or waterloo? structural alignment and the development of connectionist models of analogy. In S. J. Hanson, J. D. Cowan, and C. L. Giles, editors, *Advances in Neural Information Processing Systems 5*, pages 855–862. Morgan Kaufman Publishers, San Mateo, CA, 1993.

[4] C. M. Gray and W. Singer. Stimulus specific neuronal oscillations in orientation columns of cat visual cortex. *Proceedings of the National Academy of Sciences, USA*, 86:1698–1702, 1989.

[5] J. E. Hummel and I. Biederman. Dynamic binding in a neural network for shape recognition. *Psychological Review*, 99:480–517, 1992.

[6] D.R.J. Laming. *Information theory of choice reaction time*. Oxford University Press, New York, 1968.

[7] B. C. Love. Asynchronous connectionist binding. (Under Review), 1998.

[8] Y. Miyashita and H. S. Chang. Neuronal correlate of pictorial short-term memory in primate temporal cortex. *Nature*, 331:68–70, 1988.

[9] J. Pollack. Recursive distributed representations. *Artificial Intelligence*, 46:77–105, 1990.

[10] W. Rall. Dendritic locations of synapses and possible mechanisms for the monosynaptic EPSP in motorneurons. *Journal of Neurophysiology*, 30:1169–1193, 1967.

[11] R. Ratcliff and G. McKoon. Speed and accuracy in the processing of false statements about semantic information. *Journal of Experimental Psychology: Learning, Memory, & Cognition*, 8:16–36, 1989.

[12] L. Shastri and V. Ajjanagadde. From simple associations to systematic reasoning: A connectionist representation of rules, variables, and dynamic binding using temporal synchrony. *Behavioral and Brain Sciences*, 16:417–494, 1993.

[13] W. Softky. Fine analog coding minimizes information transmission. *Neural Networks*, 9:15–24, 1996.

[14] A. C. Tang and T. J. Sejnowski. An ecological approach tp the neural code. In *Proceedings of the Nineteenth Annual Conference of the Cogntive Science Society*, page 852, Mahwah, NJ, 1996. Erlbaum.

[15] A. Treisman and H. Schmidt. Illusory conjunctions in the perception of objects. *Cognitive Psychology*, 14:107–141, 1982.

[16] E. Vaadia, I. Haalman, M. Abeles, and H. Bergman. Dynamics of neuronal interactions in monkey cortex in relation to behavioral events. *Nature*, 2373:515–518, 1995.

[17] C. von der Malsburg. The correlation theory of brain function. Technical Report 81-2, Max-Planck-Institut for Biophysical Chemistry, Göttingen, Germany, 1981.
